# Risk Bounds for Randomized Sample Compressed Classifiers

**Mohak Shah**
Centre for Intelligent Machines
McGill University
Montreal, QC, Canada, H3A 2A7
`mohak@cim.mcgill.ca`

## Abstract

We derive risk bounds for the randomized classifiers in Sample Compression setting where the classifier-specification utilizes two sources of information viz. the compression set and the message string. By extending the recently proposed Occam's Hammer principle to the data-dependent settings, we derive point-wise versions of the bounds on the stochastic sample compressed classifiers and also recover the corresponding classical PAC-Bayes bound. We further show how these compare favorably to the existing results.

## 1 Introduction

The Sample compression framework [Littlestone and Warmuth, 1986, Floyd and Warmuth, 1995] has resulted in an important class of learning algorithms known as sample compression algorithms. These algorithms have been shown to be competitive with the state-of-the-art algorithms such as the SVM in practice [Marchand and Shawe-Taylor, 2002, Laviolette et al., 2005]. Moreover, the approach has also resulted in practical realizable bounds and has shown significant promise in using these bounds in model selection.

On another learning theoretic front, the PAC-Bayes approach [McAllester, 1999] has shown that stochastic classifier selection can prove to be more powerful than outputing a deterministic classifier. With regard to the sample compression settings, this was further confirmed in the case of sample compressed Gibbs classifier by Laviolette and Marchand [2007]. However, the specific classifier output by the algorithm (according to a selected posterior) is generally of immediate interest since this is the classifier whose future performance is of relevance in practice. Diluting such guarantees in terms of the expectancy of the risk over the posterior over the classifier space, although gives tighter risk bounds, result in averaged statements over the expected true error.

A significant result in obtaining such guarantees for the specific randomized classifier has appeared in the form of Occam's Hammer [Blanchard and Fleuret, 2007]. It deals with bounding the performance of algorithms that result in a set output when given training data. With respect to classifiers, this results in a bound on the true risk of the randomized classifier output by the algorithm in accordance with a learned posterior over the classifier space from training data. Blanchard and Fleuret [2007] also present a PAC-Bayes bound for the data-independent settings (when the classifier space is defined independently of the training data).

Motivated by this result, we derive risk bounds for the randomized sample compressed classifiers. Note that the classifier space in the case of sample compression settings, unlike other settings, is data-dependent in the sense that it is defined upon the specification of training data.[1] The rest of

the paper is organized as follows: Section 2 provides a background on the sample compressed classifiers and establishes the context; Section 3 then states the Occam's Hammer for the data-independent settings. We then derive bounds for the randomized sample compressed classifier in Section 4 followed by showing how we can recover bounds for the sample compressed Gibbs case (classical PAC-Bayes for sample compressed classifiers) in Section 5. We conclude in Section 6.

## 2 Sample Compressed (SC) Classifiers

We consider binary classification problems where the input space $\mathcal{X}$ consists of an arbitrary subset of $\mathbb{R}^n$ and the output space $\mathcal{Y} = \{-1, +1\}$. An example $\mathbf{z} \stackrel{\text{def}}{=} (\mathbf{x}, y)$ is an input-output pair where $\mathbf{x} \in \mathcal{X}$ and $y \in \mathcal{Y}$. Sample Compression learning algorithms are characterized as follows:

Given a training set $S = \{\mathbf{z}_1, \ldots, \mathbf{z}_m\}$ of $m$ examples, the classifier $A(S)$ returned by algorithm $A$ is described entirely by two *complementary sources of information*: a subset $\mathbf{z_i}$ of $S$, called the *compression set*, and a *message string* $\sigma$ which represents the additional information needed to obtain a classifier from the compression set $\mathbf{z_i}$. Given a training set $S$, the compression set $\mathbf{z_i}$ is defined by a vector $\mathbf{i}$ of indices $\mathbf{i} \stackrel{\text{def}}{=} (i_1, i_2, \ldots, i_{|\mathbf{i}|})$ with $i_j \in \{1, \ldots, m\} \ \forall j$ and $i_1 < i_2 < \ldots < i_{|\mathbf{i}|}$ and where $|\mathbf{i}|$ denotes the number of indices present in $\mathbf{i}$. Hence, $\mathbf{z}_i$ denotes the $i$th example of $S$ whereas $\mathbf{z_i}$ denotes the subset of examples of $S$ that are pointed to by the vector of indices $\mathbf{i}$ defined above. We will use $\bar{\mathbf{i}}$ to denote the set of indices not present in $\mathbf{i}$. Hence, we have $S = \mathbf{z_i} \cup \mathbf{z_{\bar{i}}}$ for any vector $\mathbf{i} \in \mathcal{I}$ where $\mathcal{I}$ denotes the set of the $2^m$ possible realizations of $\mathbf{i}$.

Finally, a learning algorithm is a sample compression learning algorithm (that is identified solely by a compression set $\mathbf{z_i}$ and a message string $\sigma$) iff there exists a *Reconstruction Function* $\mathcal{R}$ : $(\mathcal{X} \times \mathcal{Y})^{|\mathbf{i}|} \times \mathcal{K} \longrightarrow \mathcal{H}$, associated with $A$. Here, $\mathcal{H}$ is the (data-dependent) classifier space and $\mathcal{K} \subset \mathcal{I} \times \mathcal{M}$ s.t. $\mathcal{M} = \cup_{\mathbf{i} \in \mathcal{I}} \mathcal{M}(\mathbf{i})$. That is, $\mathcal{R}$ outputs a classifier $\mathcal{R}(\sigma, \mathbf{z_i})$ when given an arbitrary compression set $\mathbf{z_i} \subseteq S$ and message string $\sigma$ chosen from the set $\mathcal{M}(\mathbf{z_i})$ of all distinct messages that can be supplied to $\mathcal{R}$ with the compression set $\mathbf{z_i}$.

We seek a tight risk bound for arbitrary reconstruction functions that holds uniformly for all compression sets and message strings. For this, we adopt the PAC setting where each example $\mathbf{z}$ is drawn according to a fixed, but unknown, probability distribution $\mathcal{D}$ on $\mathcal{X} \times \mathcal{Y}$. The true risk $R(f)$ of any classifier $f$ is defined as the probability that it misclassifies an example drawn according to $\mathcal{D}$:

$$R(f) \stackrel{\text{def}}{=} \Pr_{(\mathbf{x}, y) \sim \mathcal{D}} (f(\mathbf{x}) \neq y) = \mathbf{E}_{(\mathbf{x}, y) \sim \mathcal{D}} I(f(\mathbf{x}) \neq y)$$

where $I(a) = 1$ if predicate $a$ is true and 0 otherwise. Given a training set $S = \{\mathbf{z}_1, \ldots, \mathbf{z}_m\}$ of $m$ examples, the *empirical risk* $R_S(f)$ on $S$, of any classifier $f$, is defined according to:

$$R_S(f) \stackrel{\text{def}}{=} \frac{1}{m} \sum_{i=1}^{m} I(f(\mathbf{x}_i) \neq y_i) \stackrel{\text{def}}{=} \mathbf{E}_{(\mathbf{x}, y) \sim S} I(f(\mathbf{x}) \neq y)$$

Let $\mathbf{Z}^m$ denote the collection of $m$ random variables whose instantiation gives a training sample $S = \mathbf{z}^m = \{\mathbf{z}_1, \ldots, \mathbf{z}_m\}$. To obtain the tightest possible risk bound, we will fully exploit the fact that the distribution of classification errors is a binomial. We now discuss the generic Occam's Hammer principle (w.r.t. the classification scenario) and then go on to show how it can be applied to the sample compression setting.

## 3 Occam's Hammer for data independent setting

In this section, we briefly detail the Occam's hammer [Blanchard and Fleuret, 2007] for data-independent setting. For the sake of simplicity, we retain the key notations of Blanchard and Fleuret [2007]. Occam's hammer work by bounding the probability of bad event defined as follows. For every classifier $h \in \mathcal{H}$, and a confidence parameter $\delta \in [0, 1]$, the bad event $\mathcal{B}(h, \delta)$ is defined as the region where the desired property on the classifier $h$ does not hold, with probability $\delta$. That is, $\Pr_{S \sim \mathcal{D}^m}[S \in \mathcal{B}(h, \delta)] \leq \delta$. Further, it assumes that this region is nondecreasing in $\delta$. Intuitively, this means that with decreasing $\delta$ the bound on the true error of the classifier $h$ becomes tighter.

With the above assumption satisfied, let, $\mathbf{P}$ be a non-negative reference measure on the classifier space $\mathcal{H}$ known as the volumic measure. Let $\Pi$ be a probability distribution on $\mathcal{H}$ absolutely continuous w.r.t. $\mathbf{P}$ such that $\pi = \frac{d\Pi}{d\mathbf{P}}$. Let $\Gamma$ be a probability distribution on $(0, +\infty)$ (the inverse density prior). Then Occam's Hammer [Blanchard and Fleuret, 2007] states that:

**Theorem 1** *[Blanchard and Fleuret, 2007] Given the above assumption and $\mathbf{P}, \Pi, \Gamma$ defined as above, define the level function*

$$\Delta(h, u) = \min(\delta \pi(h)\beta(u), 1).$$

*where $\beta(x) = \int_0^x u d\Gamma(u)$ for $x \in (0, +\infty)$. Then for any algorithm $S \mapsto \theta_S$ returning a probability density $\theta_S$ over $\mathcal{H}$ with respect to $\mathbf{P}$, and such that $(S, h) \mapsto \theta_S(h)$ is jointly measurable in its two variables, it holds that*

$$\Pr_{S \sim \mathcal{D}^m, h \sim \mathbf{Q}} \left[ S \in \mathcal{B}(h, \Delta(h, \theta_S(h)^{-1})) \right] \leq \delta,$$

*where $\mathbf{Q}$ is the distribution on $\mathcal{H}$ such that $\frac{d\mathbf{Q}}{d\mathbf{P}} = \theta_S$.*

Note above that $\mathbf{Q}$ is the (data-dependent) posterior distribution on $\mathcal{H}$ after observing the data sample $S$ while $\mathbf{P}$ is the data-independent prior on $\mathcal{H}$. The subscript $S$ in $\theta_S$ denotes this. Moreover, the distribution $\Pi$ on the space of classifiers may or may not be data-dependent. As we will see later, in the case of sample compression learning settings we will consider priors over the space of classifiers without reference to the data (such as PAC-Bayes case). To this end, we can either opt for a prior $\Pi$ independent of the data or make it the same as the volume measure $\mathbf{P}$ which establishes a distribution on the classifier space without reference to the data.

## 4 Bounds for Randomized SC Classifiers

We work in the sample compression settings and as mentioned before, each classifier in this setting is denoted in terms of a compression set and a message string. A reconstruction function then uses these two information sources to reconstruct the classifier. This essentially means that we deal with a data-dependent hypothesis space. This is in contrast with other notions of hypothesis class complexity measures such as VC dimension. The hypothesis space is defined, in our case, based on the size of data sample (and not the actual contents of the sample). Hence, we consider the priors built on the size of the possible compression sets and associated message strings. More precisely, we consider prior distribution $\mathbf{P}$ with probability density $P(\mathbf{z_i}, \sigma)$ to be facotorizable in its compression set dependent component and message string component (conditioned on a given compression set) such that:

$$P(\mathbf{z_i}, \sigma) = P_{\mathcal{I}}(\mathbf{i}) P_{\mathcal{M}(\mathbf{z_i})}(\sigma) \tag{1}$$

with $P_{\mathcal{I}}(\mathbf{i}) = \frac{1}{\binom{m}{|\mathbf{i}|}} p(|\mathbf{i}|)$ such that $\sum_{d=0}^m p(d) = 1$. The above choice of the form for $P_{\mathcal{I}}(\mathbf{i})$ is appropriate since we do not have any *a priori* information to distinguish one compression set from other. However, as we will see later, we should choose $p(d)$ such that we give more weight to smaller compression sets.

Let $\mathcal{P}_{\mathcal{K}}$ be the set of all distributions $P$ on $\mathcal{K}$ satisfying above equation. Then, we are interested in algorithms that output a posterior $Q \in \mathcal{P}_{\mathcal{K}}$ over the space of classifiers with probability density $Q(\mathbf{z_i}, \sigma)$ factorizable as $Q_{\mathcal{I}}(\mathbf{i}) Q_{\mathcal{M}(\mathbf{z_i})}(\sigma)$. A sample compressed classifier is then defined by choosing a classifier $(\mathbf{z_i}, \sigma)$ according to the posterior $Q(\mathbf{z_i}, \sigma)$. This is basically the Gibbs classifier defined in the PAC-Bayes settings where the idea is to bound the true risk of this Gibbs classifier defined as $R(G_Q) = \mathbf{E}_{(\mathbf{z_i}, \sigma) \sim Q} R((\mathbf{z_i}, \sigma))$. On the other hand, we are interested in bounding the true risk of the specific classifier $(\mathbf{z_i}, \sigma)$ output according to $Q$. As shown in [Laviolette and Marchand, 2007], a rescaled posterior $\overline{Q}$ of the following form can provide tighter guarantees while maintaining the Occam's principle of parsimony.

**Definition 2** *Given a distribution $Q \in \mathcal{P}_{\mathcal{K}}$, we denote by $\overline{Q}$ the distribution:*

$$\overline{Q}(\mathbf{z_i}, \sigma) \stackrel{\text{def}}{=} \frac{Q(\mathbf{z_i}, \sigma)}{|\bar{\mathbf{i}}| \mathbf{E}_{(\mathbf{z_i}, \sigma) \sim Q} \frac{1}{|\bar{\mathbf{i}}|}} = \frac{Q_{\mathcal{I}}(\mathbf{i}) Q_{\mathcal{M}(\mathbf{z_i})}(\sigma)}{|\bar{\mathbf{i}}| \mathbf{E}_{(\mathbf{z_i}, \sigma) \sim Q} \frac{1}{|\bar{\mathbf{i}}|}} = \overline{Q_{\mathcal{I}}}(\mathbf{i}) Q_{\mathcal{M}(\mathbf{z_i})}(\sigma) \qquad \forall (\mathbf{z_i}, \sigma) \in \mathcal{K}$$

Hence, note that the posterior is effectively rescaled for the compression set part. Hence, any classifier $(\mathbf{z_i}, \sigma) \sim \overline{Q} = \mathbf{i} \sim \overline{Q_{\mathcal{I}}}, \sigma \sim Q_{\mathcal{M}(\mathbf{z_i})}$. Further, if we denote by $d_{\overline{Q}}$ the expected value of the compression set size over the choice of parameters according to the scaled posterior, $d_{\overline{Q}} \stackrel{\text{def}}{=} \mathbf{E}_{\mathbf{i} \sim \overline{Q_{\mathcal{I}}}, \sigma \sim Q_{\mathcal{M}(\mathbf{z_i})}} |\mathbf{i}|$, then,

$$\mathbf{E}_{(\mathbf{z_i}, \sigma) \sim Q} \frac{1}{|\bar{\mathbf{i}}|} = \frac{1}{\mathbf{E}_{\mathbf{i} \sim \overline{Q_{\mathcal{I}}}, \sigma \sim Q_{\mathcal{M}(\mathbf{z_i})}} |\bar{\mathbf{i}}|} = \frac{1}{m - d_{\overline{Q}}}$$

Now, we proceed to derive the bounds for the randomized sample compressed classifiers starting with a PAC-Bayes bound.

## 4.1 A PAC-Bayes Bound for randomized SC classifier

We exploit the fact that the distribution of the errors is binomial and define the following error quantities (for a given $\mathbf{i}$, and hence $\mathbf{z_i}$ over $\mathbf{z}_{|\bar{\mathbf{i}}|}$):

**Definition 3** *Let $S \in \mathcal{D}^m$ with $\mathcal{D}$ a distribution on $\mathcal{X} \times \mathcal{Y}$, and $(\mathbf{z_i}, \sigma) \in \mathcal{K}$. We denote by $\mathrm{Bin}_S(\mathbf{i}, \sigma)$, the probability that the classifier $\mathcal{R}(\mathbf{z_i}, \sigma)$ of (true) risk $R(zb_{\mathbf{i}}, \sigma)$ makes $|\bar{\mathbf{i}}| R_{\mathbf{z}_{\bar{\mathbf{i}}}}(\mathbf{z_i}, \sigma)$ or fewer errors on $\mathbf{z'_{\bar{\mathbf{i}}}} \sim \mathcal{D}^{|\bar{\mathbf{i}}|}$. That is,*

$$\mathrm{Bin}_S(\mathbf{i}, \sigma) = \sum_{\lambda=0}^{|\bar{\mathbf{i}}| R_{\mathbf{z}_{\bar{\mathbf{i}}}}(\mathbf{z_i}, \sigma)} \binom{|\bar{\mathbf{i}}|}{\lambda} (R(\sigma, \mathbf{z_i}))^{\lambda} (1 - R(\sigma, \mathbf{z_i}))^{|\bar{\mathbf{i}}| - \lambda}$$

*and by $B_S(\mathbf{i}, \sigma)$, the probability that this classifier makes exactly $|\bar{\mathbf{i}}| R_{\mathbf{z}_{\bar{\mathbf{i}}}}(\mathbf{z_i}, \sigma)$ errors on $\mathbf{z'_{\bar{\mathbf{i}}}} \sim \mathcal{D}^{|\bar{\mathbf{i}}|}$. That is,*

$$B_S(\mathbf{i}, \sigma) = \binom{|\bar{\mathbf{i}}|}{|\bar{\mathbf{i}}| R_{\mathbf{z}_{\bar{\mathbf{i}}}}(\mathbf{z_i}, \sigma)} (R(\mathbf{z_i}, \sigma))^{|\bar{\mathbf{i}}| R_{\mathbf{z}_{\bar{\mathbf{i}}}}(\mathbf{z_i}, \sigma)} (1 - R(\mathbf{z_i}, \sigma))^{|\bar{\mathbf{i}}| - |\bar{\mathbf{i}}| R_{\mathbf{z}_{\bar{\mathbf{i}}}}(\mathbf{z_i}, \sigma)}$$

Now, approximating the binomial by relative entropy Chernoff bound [Langford, 2005], we have, for a classifier $f$:

$$\sum_{j=0}^{mR_S(f)} \binom{m}{j} (R(f))^j (1 - R(f))^{m-j} \leq \exp(-m \cdot \mathrm{kl}(R_S(f) \| R(f)))$$

for all $R_S(f) \leq R(f)$.

As also shown in [Laviolette and Marchand, 2007], since $\binom{m}{j} = \binom{m}{m-j}$ and $\mathrm{kl}(R_S(f) \| R(f)) = \mathrm{kl}(1 - R_S(f) \| 1 - R(f))$, the above inequality holds true for each factor inside the sum on the left hand side. Consequently, in the case of sample compressed classifier, $\forall (\mathbf{z_i}, \sigma) \in \mathcal{K}$ and $\forall S \in (\mathcal{X} \times \mathcal{Y})^m$:

$$B_S(\mathbf{i}, \sigma) \leq \exp \left[ -|\bar{\mathbf{i}}| \cdot \mathrm{kl}(R_{\mathbf{z}_{\bar{\mathbf{i}}}}(\sigma, \mathbf{z_i}) \| R(\sigma, \mathbf{z_i})) \right] \tag{2}$$

Bounding this by $\delta$ yields:

$$\Pr_{S \sim \mathcal{D}^m} \left( \mathrm{kl}(R_{\mathbf{z}_{\bar{\mathbf{i}}}}(\sigma, \mathbf{z_i}) \| R(\sigma, \mathbf{z_i})) \leq \frac{\ln \frac{1}{\delta}}{|\bar{\mathbf{i}}|} \right) \geq 1 - \delta \tag{3}$$

Now, consider the quantity in the probability in Equation 3 as the bad event over classifiers defined by a compression set $\mathbf{i}$ and an associated message string $\sigma$. Let $\psi_{\mathbf{z}^m}(\mathbf{i}, \sigma)$ be the posterior probability density of the rescaled data-dependent posterior distribution $\overline{Q}$ over the classifier space *with respect to* the volume measure $\mathbf{P}$. We can now replace $\delta$ for this bad event by the delta of the Occam's hammer defined as:

$$
\begin{aligned}
\ln(\min(\delta \pi(h_S) \beta(\psi_{\mathbf{z}^m}(\mathbf{i}, \sigma)^{-1}), 1)^{-1}) &= \ln_+ \left( \frac{1}{\delta \cdot \pi(h)} \cdot \frac{1}{\min((k+1)^{-1} \psi_{\mathbf{z}^m}(\mathbf{i}, \sigma)^{-\frac{k+1}{k}}, 1)} \right) \\
&= \ln_+ \left( \frac{1}{\delta \cdot \pi(h)} \cdot \max((k+1) \psi_{\mathbf{z}^m}(\mathbf{i}, \sigma)^{\frac{k+1}{k}}, 1) \right) \\
&\leq \ln_+ \left( \frac{1}{\delta \cdot \pi(h)} \cdot (k+1) \max(\psi_{\mathbf{z}^m}(\mathbf{i}, \sigma)^{\frac{k+1}{k}}, 1) \right) \\
&\leq \ln \left( \frac{1}{\delta \cdot \pi(h)} \cdot (k+1) \right) + \ln_+ \left( \psi_{\mathbf{z}^m}(\mathbf{i}, \sigma)^{\frac{k+1}{k}} \right)
\end{aligned}
$$

where $\ln_+$ denotes $\max(0, \ln)$, the positive part of the logarithm.

However, note that we are interested in data-independent priors over the space of classifiers[2], and hence, we consider our prior $\Pi$ to be the same as the volume measure $\mathbf{P}$ over the classifier space yielding $\pi$ as unity. That is, our prior gives a distribution over the classifier space without any regard to the data. Substituting for $\psi_{\mathbf{z}^m}(\mathbf{i}, \sigma)$ (the fraction of respective densities; Radon-Nikodym derivative)[3], we obtain the following result:

**Theorem 4** *For any reconstruction function $\mathcal{R} : \mathcal{D}^m \times \mathcal{K} \longrightarrow \mathcal{H}$ and for any prior distribution $\mathbf{P}$ over compression set and message strings, the sample compression algorithms $A(S)$ returns a posterior distribution $Q$, then, for $\delta \in (0, 1]$ and $k > 0$, we have:*

$$
\Pr_{S \sim \mathcal{D}^m, \mathbf{i} \sim \overline{Q_{\mathcal{I}}}, \sigma \sim Q_{\mathcal{M}(\mathbf{z_i})}} \left[ \mathrm{kl}(R_{\mathbf{z_{\bar{i}}}}(\mathbf{z_i}, \sigma) \| R(\mathbf{z_i}, \sigma)) \right.
$$
$$
\left. \leq \frac{1}{m - |\mathbf{i}|} \left[ \ln\left(\frac{k+1}{\delta}\right) + \left(1 + \frac{1}{k}\right) \ln_+ \left(\frac{\overline{Q}(\mathbf{z_i}, \sigma)}{P(\mathbf{z_i}, \sigma)}\right) \right] \right] \geq 1 - \delta
$$

*where $R_{\mathbf{z_{\bar{i}}}}(\mathbf{z_i}, \sigma)$ is the empirical risk of the classifier reconstructed from $(\mathbf{z_i}, \sigma)$ on the training examples not in the compression set and $R(\mathbf{z_i}, \sigma)$ is the corresponding true risk.*

Note that we do not encounter the $\frac{1}{m - d_{\overline{Q}}}$ factor in the bound instead of $\frac{1}{m - |\mathbf{i}|}$ unlike the bound of Laviolette and Marchand [2007]. This is because the PAC-Bayes bound of Laviolette and Marchand [2007] computes the *expectancy* over the kl-divergence of the empirical and true risk of the classifiers chosen according to $\overline{Q}$. This, as a result of rescaling of $Q$ in preference of smaller compression sets, is reflected in the bound. On the other hand, the bound of Theorem 4 is a point-wise version bounding the true error of *the specific classifier chosen* according to $\overline{Q}$ and hence concerns the specific compression set utilized by this classifier.

## 4.2 A Binomial Tail Inversion Bound for randomized SC classifier

A tighter condition can be imposed on the true risk of the classifier by considering the binomial tail inversion over the distribution of errors. The *binomial tail inversion* $\overline{\mathrm{Bin}}\left(\frac{k}{m}, \delta\right)$ is defined as the largest risk value that a classifier can have while still having a probability of at least $\delta$ of observing at most $k$ errors out of $m$ examples:

$$
\overline{\mathrm{Bin}}\left(\frac{k}{m}, \delta\right) \overset{\text{def}}{=} \sup\left\{ r : \mathrm{Bin}\left(\frac{k}{m}, r\right) \geq \delta \right\}
$$

where

$$
\mathrm{Bin}\left(\frac{k}{m}, r\right) \overset{\text{def}}{=} \sum_{j=0}^{k} \binom{m}{j} r^j (1 - r)^{m - j}
$$

From this definition, it follows that $\overline{\mathrm{Bin}}\left(R_S(f), \delta\right)$ is the *smallest* upper bound, which holds with probability at least $1 - \delta$, on the true risk of any classifier $f$ with an observed empirical risk $R_S(f)$ on a test set of $m$ examples (test set bound):

$$
\mathbf{P}_{\mathbf{Z}^m} \left\{ R(f) \leq \overline{\mathrm{Bin}}\left(R_{\mathbf{Z}^m}(f), \delta\right) \right\} \geq 1 - \delta \quad \forall f \tag{4}
$$

This bound can be converted to a training set bound in a standard manner by considering a measure over the classifier space (see for instance [Langford, 2005, Theorem 4.1]). Moreover, in the sample compression case, we are interested in the empirical risk of the classifier on the examples not in the compression set (consistent compression set assumption). Now, let $\delta_r$ be a $\delta$-weighed measure on the classifier space, i.e., $\mathbf{i}$ and $\sigma$. Then, for the compression sets and associated message strings,

consider the following bad event with empirical risk of the classifier measured as $\mathrm{Bin}_S((\mathbf{z_i}, \sigma))$ for $\mathbf{i} \sim \overline{Q_{\mathcal{I}}}, \sigma \sim Q_{\mathcal{M}(\mathbf{z_i})}$:

$$\mathcal{B}(h, \delta) = \left\{ R(\mathbf{z_i}, \sigma) > \overline{\mathrm{Bin}}(R_{\mathbf{z_{\bar{i}}}}(\mathbf{z_i}, \sigma), \delta_r) \right\}$$

Now, we replace $\delta_r$ with the level function of Occam's hammer (with the same assumption of $\Pi = \mathbf{P}, \pi = 1$):

$$
\begin{aligned}
\min(\delta \pi(h_S) \beta(\psi_{\mathbf{z}^m}(\mathbf{i}, \sigma)^{-1}), 1) \quad &\leq \quad \delta \cdot \min((k+1)^{-1} \psi_{\mathbf{z}^m}(\mathbf{i}, \sigma)^{-\frac{k+1}{k}}, 1) \\
&\leq \quad \delta \cdot \frac{1}{\max((k+1)\psi_{\mathbf{z}^m}(\mathbf{i}, \sigma)^{\frac{k+1}{k}}, 1)} \\
&\leq \quad \delta \frac{1}{(k+1)\max(\psi_{\mathbf{z}^m}(\mathbf{i}, \sigma)^{\frac{k+1}{k}}, 1)} \\
&\leq \quad \frac{\delta}{(k+1)\psi_{\mathbf{z}^m}(\mathbf{i}, \sigma)^{\frac{k+1}{k}}}
\end{aligned}
$$

Hence, we have proved the following:

**Theorem 5** *For any reconstruction function $\mathcal{R} : \mathcal{D}^m \times \mathcal{K} \longrightarrow \mathcal{H}$ and for any prior distribution $\mathbf{P}$ over the compression set and message strings, the sample compression algorithms $A(S)$ returns a posterior distribution $Q$, then, for $\delta \in (0, 1]$ and $k > 0$, we have:*

$$\Pr_{S \sim \mathcal{D}^m, \mathbf{i} \sim \overline{Q_{\mathcal{I}}}, \sigma \sim Q_{\mathcal{M}(\mathbf{z_i})}} \left[ R(\mathbf{z_i}, \sigma) \leq \overline{\mathrm{Bin}}\left( R_{\mathbf{z_{\bar{i}}}}(\mathbf{z_i}, \sigma), \frac{\delta}{(k+1)\left(\frac{\overline{Q}(\mathbf{z_i}, \sigma)}{P(\mathbf{z_i}, \sigma)}\right)^{\frac{k+1}{k}}} \right) \right] \geq 1 - \delta$$

We can obtain a looser bound by approximating the binomial tail inversion bound using [Laviolette et al., 2005, Lemma 1]:

**Corollary 6** *Given all our previous definitions, the following holds with probability $1 - \delta$ over the joint draw of $S \sim \mathcal{D}^m$ and $\mathbf{i} \sim \overline{Q_{\mathcal{I}}}, \sigma \sim Q_{\mathcal{M}(\mathbf{z_i})}$:*

$$
\begin{aligned}
R(\mathbf{z_i}, \sigma) \leq 1 - \exp\left( \frac{-1}{m - |\mathbf{i}| - |\bar{\mathbf{i}}| R_{\mathbf{z_{\bar{i}}}}(\mathbf{z_i}, \sigma)} \left[ \ln\left( \frac{m - |\mathbf{i}|}{|\bar{\mathbf{i}}| R_{\mathbf{z_{\bar{i}}}}(\mathbf{z_i}, \sigma)} \right) + \ln\left( \frac{k+1}{\delta} \right) \right.\right. \\
\left.\left. + (1 + \frac{1}{k}) \ln\left( \frac{\overline{Q}(\mathbf{z_i}, \sigma)}{P(\mathbf{z_i}, \sigma)} \right) \right] \right)
\end{aligned}
$$

## 5  Recovering the PAC-Bayes bound for SC Gibbs Classifier

Let us now see how a bound can be obtained for the Gibbs setting. We follow the general line of argument of Blanchard and Fleuret [2007] to recover the PAC-Bayes bound for the Sample Compressed Gibbs classifier. However, note that we do this for the data-dependent setting here and also utilize the rescaled posterior over the space of sample compressed classifiers.

The PAC-Bayes bound of Theorem 4 basically states that

$$\mathbf{E}_{S \sim \mathcal{D}^m}\left[ \Pr_{\mathbf{i} \sim \overline{Q_{\mathcal{I}}}, \sigma \sim Q_{\mathcal{M}(\mathbf{z_i})}} [\mathrm{kl}(R_{\mathbf{z_{\bar{i}}}}(\mathbf{z_i}, \sigma) \| R(\mathbf{z_i}, \sigma)) > \varphi(\delta)] \right] \leq \delta$$

where

$$\varphi(\delta) = \frac{1}{m - |\mathbf{i}|} \left[ \ln\left( \frac{k+1}{\delta} \right) + (1 + \frac{1}{k}) \ln_+ \left( \frac{\overline{Q}(\mathbf{z_i}, \sigma)}{P(\mathbf{z_i}, \sigma)} \right) \right]$$

Consequently,

$$\mathbf{E}_{S \sim \mathcal{D}^m}\left[ \Pr_{\mathbf{i} \sim \overline{Q_{\mathcal{I}}}, \sigma \sim Q_{\mathcal{M}(\mathbf{z_i})}} [\mathrm{kl}(R_{\mathbf{z_{\bar{i}}}}(\mathbf{z_i}, \sigma) \| R(\mathbf{z_i}, \sigma)) > \varphi(\delta\gamma)] \right] \leq \delta\gamma$$

Now, bounding the argument of expectancy above using the Markov inequality, we get:

$$\Pr_{S \sim \mathcal{D}^m}\left[ \Pr_{\mathbf{i} \sim \overline{Q_{\mathcal{I}}}, \sigma \sim Q_{\mathcal{M}(\mathbf{z_i})}} [\mathrm{kl}(R_{\mathbf{z_{\bar{i}}}}(\mathbf{z_i}, \sigma) \| R(\mathbf{z_i}, \sigma)) > \varphi(\delta\gamma)] > \gamma \right] \leq \delta$$

Now, discretizing the argument over $(\delta_i, \gamma_i) = (\delta 2^{-i}, 2^{-i})$, we obtain

$$\Pr_{S \sim \mathcal{D}^m}\left[\Pr_{\mathbf{i} \sim \overline{Q_{\mathcal{I}}}, \sigma \sim Q_{\mathcal{M}(\mathbf{z_i})}}[\mathrm{kl}(R_{\mathbf{z_{\bar{i}}}}(\mathbf{z_i}, \sigma) \| R(\mathbf{z_i}, \sigma)) > \varphi(\delta_i \gamma_i)] > \gamma_i\right] \leq \delta_i$$

Taking the union bound over $\delta_i, i \geq 1$ now yields:

$$\Pr_{S \sim \mathcal{D}^m}\left[\Pr_{\mathbf{i} \sim \overline{Q_{\mathcal{I}}}, \sigma \sim Q_{\mathcal{M}(\mathbf{z_i})}}[\mathrm{kl}(R_{\mathbf{z_{\bar{i}}}}(\mathbf{z_i}, \sigma) \| R(\mathbf{z_i}, \sigma)) > \varphi(\delta 2^{-2i}] \leq 2^{-i}\right] > 1 - \delta \quad \forall i \geq 0$$

Now, let us consider the argument of the above statement for a fixed sample $S$. Then, for all $i \geq 0$, the following holds with probability $1 - \delta$:

$$\Pr_{\mathbf{i} \sim \overline{Q_{\mathcal{I}}}, \sigma \sim Q_{\mathcal{M}(\mathbf{z_i})}}\left[\mathrm{kl}(R_{\mathbf{z_{\bar{i}}}}(\mathbf{z_i}, \sigma) \| R(\mathbf{z_i}, \sigma)) > \frac{1}{m - |\mathbf{i}|}\left[\ln\left(\frac{k+1}{\delta}\right) + 2i \ln 2 \right.\right.$$
$$\left.\left. + (1 + \frac{1}{k}) \ln_+ \left(\frac{\overline{Q}(\mathbf{z_i}, \sigma)}{P(\mathbf{z_i}, \sigma)}\right)\right]\right] \leq 2^{-i}$$

and hence:

$$\Pr_{\mathbf{i} \sim \overline{Q_{\mathcal{I}}}, \sigma \sim Q_{\mathcal{M}(\mathbf{z_i})}}\left[\Phi^S(\mathbf{z_i}, \sigma) > 2i \ln 2\right] \leq 2^{-i}$$

where:

$$\Phi^S(\mathbf{z_i}, \sigma) = (m - |\mathbf{i}|)\mathrm{kl}(R_{\mathbf{z_{\bar{i}}}}(\mathbf{z_i}, \sigma) \| R(\mathbf{z_i}, \sigma)) - \ln\left(\frac{k+1}{\delta}\right) - (1 + \frac{1}{k}) \ln_+ \left(\frac{\overline{Q}(\mathbf{z_i}, \sigma)}{P(\mathbf{z_i}, \sigma)}\right)$$

We wish to bound, for the Gibbs classifier, $\mathbf{E}_{\mathbf{i} \sim \overline{Q_{\mathcal{I}}}, \sigma \sim Q_{\mathcal{M}(\mathbf{z_i})}} \Phi^S(\mathbf{z_i}, \sigma)$:

$$\mathbf{E}_{\mathbf{i} \sim \overline{Q_{\mathcal{I}}}, \sigma \sim Q_{\mathcal{M}(\mathbf{z_i})}}[\Phi^S(\mathbf{z_i}, \sigma)] \leq \int_{2i \ln 2 > 0} \Pr_{\mathbf{i} \sim \overline{Q_{\mathcal{I}}}, \sigma \sim Q_{\mathcal{M}(\mathbf{z_i})}}[\Phi^S(\mathbf{z_i}, \sigma) \geq 2i \ln 2] d(2i \ln 2)$$
$$\leq 2 \ln 2 \sum_{i \geq 0} \Pr_{\mathbf{i} \sim \overline{Q_{\mathcal{I}}}, \sigma \sim Q_{\mathcal{M}(\mathbf{z_i})}}[\Phi^S(\mathbf{z_i}, \sigma) \geq 2i \ln 2] \leq 3 \quad (5)$$

Now, we have:

**Lemma 7** *[Laviolette and Marchand, 2007] For any $f : \mathcal{K} \longrightarrow \mathbb{R}^+$, and for any $Q, Q' \in \mathcal{P}_{\mathcal{K}}$ related by*

$$Q'(\mathbf{z_i}, \sigma) f(\mathbf{z_i}, \sigma) = \frac{1}{\mathbf{E}_{(\mathbf{z_i}, \sigma) \sim Q} \frac{1}{f(\mathbf{z_i}, \sigma)}} Q(\mathbf{z_i}, \sigma),$$

*we have:*

$$\mathbf{E}_{(\mathbf{z_i}, \sigma) \sim Q'}\left(f(\mathbf{z_i}, \sigma) \mathrm{kl}(R_{\mathbf{z_{\bar{i}}}}(\mathbf{z_i}, \sigma) \| R(\mathbf{z_i}, \sigma))\right) \geq \frac{1}{\mathbf{E}_{(\mathbf{z_i}, \sigma) \sim Q}\left(\frac{1}{f(\mathbf{z_i}, \sigma)}\right)} \mathrm{kl}(R_S(G_Q) \| R(G_Q))$$

*where $R_S(G_Q)$ and $R(G_Q)$ denote the empirical and true risk of the Gibbs classifier with posterior $Q$ respectively.*

Hence, with $Q' = \overline{Q}$ and $f(\mathbf{z_i}, \sigma) = |\bar{\mathbf{i}}|$, Lemma 7 yields:

$$\mathbf{E}_{(\mathbf{z_i}, \sigma) \sim \overline{Q}}(|\bar{\mathbf{i}}| \mathrm{kl}(R_{\mathbf{z_{\bar{i}}}}(\mathbf{z_i}, \sigma) \| R(\mathbf{z_i}, \sigma))) \geq \frac{1}{\frac{1}{m - d_{\overline{Q}}}} \mathrm{kl}(R_S(G_Q) \| R(G_Q)) \quad (6)$$

Further,

$$\mathbf{E}_{\mathbf{i} \sim \overline{Q_{\mathcal{I}}}, \sigma \sim Q_{\mathcal{M}(\mathbf{z_i})}}\left[\ln_+ \frac{\overline{Q}(\mathbf{z_i}, \sigma)}{P(\mathbf{z_i}, \sigma)}\right] = \mathbf{E}_{\mathbf{i} \sim \overline{Q_{\mathcal{I}}}, \sigma \sim Q_{\mathcal{M}(\mathbf{z_i})}}\left[\ln_+ \left(\frac{\overline{Q}(\mathbf{z_i}, \sigma)}{P_{\mathcal{I}}(\mathbf{i}) P_{\mathcal{M}(\mathbf{z_i})}(\sigma)}\right)\right]$$
$$= \mathbf{E}_{(\mathbf{z_i}, \sigma) \sim P}\left[\left(\frac{\overline{Q}(\mathbf{z_i}, \sigma)}{P_{\mathcal{I}}(\mathbf{i}) P_{\mathcal{M}(\mathbf{z_i})}(\sigma)}\right) \cdot \ln_+ \left(\frac{\overline{Q}(\mathbf{z_i}, \sigma)}{P_{\mathcal{I}}(\mathbf{i}) P_{\mathcal{M}(\mathbf{z_i})}(\sigma)}\right)\right]$$
$$\leq \mathbf{E}_{(\mathbf{z_i}, \sigma) \sim P}\left[\left(\frac{\overline{Q}(\mathbf{z_i}, \sigma)}{P_{\mathcal{I}}(\mathbf{i}) P_{\mathcal{M}(\mathbf{z_i})}(\sigma)}\right) \cdot \ln \left(\frac{\overline{Q}(\mathbf{z_i}, \sigma)}{P_{\mathcal{I}}(\mathbf{i}) P_{\mathcal{M}(\mathbf{z_i})}(\sigma)}\right)\right]$$
$$- \max_{0 \leq x < 1} x \ln x$$
$$\leq \mathrm{KL}(\overline{Q} \| P) + 0.5 \quad (7)$$

Equations 6 and 7 along with Equation 5 and substituting $k = m - 1$ yields the final result:

**Theorem 8** *For any reconstruction function* $\mathcal{R} : \mathcal{D}^m \times \mathcal{K} \longrightarrow \mathcal{H}$ *and for any prior distribution* **P** *over compression set and message strings, for* $\delta \in (0, 1]$, *we have:*

$$
\Pr_{S \sim \mathcal{D}^m} \Bigg( \forall Q \in \mathcal{P}_{\mathcal{K}} : \mathrm{kl}(R_S(G_Q) \| R(G_Q))
$$
$$
\leq \frac{1}{m - d_{\overline{Q}}} \Bigg[ (1 + \frac{1}{m-1}) \mathrm{KL}(\overline{Q} \| P) + \frac{1}{2(m-1)} + \ln\left(\frac{m}{\delta}\right) + 3.5 \Bigg] \Bigg) \geq 1 - \delta
$$

Theorem 8 recovers almost exactly the PAC-Bayes bound for the Sample Compressed Classifiers of Laviolette and Marchand [2007]. The key differences are an additional $\frac{1}{(m - d_{\overline{Q}})(m-1)}$ weighted KL-divergence term, $\ln(\frac{m}{\delta})$ instead of the $\ln(\frac{m+1}{\delta})$ and the additional trailing terms bounded by $\frac{4}{m - d_{\overline{Q}}}$. Note that the bound of Theorem 8 is derived in a relatively more straightforward manner with the Occam's Hammer criterion.

## 6 Conclusion

It has been shown that stochastic classifier selection is preferable to deterministic selection by the PAC-Bayes principle resulting in tighter risk bounds over averaged risk of classifiers according to the learned posterior. Further, this observation resulted in tight bounds in the case of stochastic sample compressed classifiers [Laviolette and Marchand, 2007] also showing that sparsity considerations are of importance even in this scenario via. the rescaled posterior. However, of immediate relevance are the guarantees of the specific classifier output by such algorithms according to the learned posterior and hence a point-wise version of this bound is indeed needed. We have derived bounds for such randomized sample compressed classifiers by adapting Occam's Hammer principle to the data-dependent sample compression settings. This has resulted in bounds on the specific classifier output by a sample compression learning algorithm according to the learned data-dependent posterior and is more relevant in practice. Further, we also showed how classical PAC-Bayes bound for the sample compressed Gibbs classifier can be recovered in a more direct manner and show that this compares favorably to the existing result of Laviolette and Marchand [2007].

#### Acknowledgments

The author would like to thank John Langford for interesting discussions.

## Footnotes

[1]Note that the classifier space depends on the amount of the training data as we see further and not on the training data themselves. Hence, a data-independent prior over the classifier space can still be obtained in this setting, e.g., in the PAC-Bayes case, owing to the independence of the classifier space definition from the content of the training data.

[2]Hence, the missing $S$ in the subscript of $\pi(h)$ in the r.h.s. above.

[3]Alternatively, let $P(\mathbf{z_i}, \sigma)$ and $\overline{Q}(\mathbf{z_i}, \sigma)$ denote the probability densities of the prior distribution $P$ and rescaled posterior distributions $\overline{Q}$ over classifiers such that $d\overline{Q} = \overline{Q}(\mathbf{z_i}, \sigma)d\mu$ and $dP = P(\mathbf{z_i}, \sigma)d\mu$ w.r.t. some measure $\mu$. This too yields $\frac{d\overline{Q}}{dP} = \frac{\overline{Q}(\mathbf{z_i}, \sigma)}{P(\mathbf{z_i}, \sigma)}$. Note that the final expression is independent of the underlying measure $\mu$.

## References

Gilles Blanchard and François Fleuret. Occam's hammer. In *Proceedings of the 20th Annual Conference on Learning Theory (COLT-2007)*, volume 4539 of *Lecture Notes on Computer Science*, pages 112–126, 2007.

Sally Floyd and Manfred Warmuth. Sample compression, learnability, and the Vapnik-Chervonenkis dimension. *Machine Learning*, 21(3):269–304, 1995.

John Langford. Tutorial on practical prediction theory for classification. *Journal of Machine Learning Research*, 3:273–306, 2005.

François Laviolette and Mario Marchand. PAC-Bayes risk bounds for stochastic averages and majority votes of sample-compressed classifiers. *Journal of Machine Learning Research*, 8:1461–1487, 2007.

Francois Laviolette, Mario Marchand, and Mohak Shah. Margin-sparsity trade-off for the set covering machine. In *Proceedings of the 16th European Conference on Machine Learning, ECML 2005*, volume 3720 of *Lecture Notes in Artificial Intelligence*, pages 206–217. Springer, 2005.

N. Littlestone and M. Warmuth. Relating data compression and learnability. Technical report, University of California Santa Cruz, Santa Cruz, CA, 1986.

Mario Marchand and John Shawe-Taylor. The Set Covering Machine. *Journal of Machine Learning Reasearch*, 3:723–746, 2002.

David McAllester. Some PAC-Bayesian theorems. *Machine Learning*, 37:355–363, 1999.
